# Maximum likelihood trajectories for continuous-time Markov chains

**Theodore J. Perkins**
Ottawa Hospital Research Institute
Ottawa, Ontario, Canada
tperkins@ohri.ca

## Abstract

Continuous-time Markov chains are used to model systems in which transitions between states as well as the time the system spends in each state are random. Many computational problems related to such chains have been solved, including determining state distributions as a function of time, parameter estimation, and control. However, the problem of inferring most likely trajectories, where a trajectory is a sequence of states as well as the amount of time spent in each state, appears unsolved. We study three versions of this problem: (i) an initial value problem, in which an initial state is given and we seek the most likely trajectory until a given final time, (ii) a boundary value problem, in which initial and final states and times are given, and we seek the most likely trajectory connecting them, and (iii) trajectory inference under partial observability, analogous to finding maximum likelihood trajectories for hidden Markov models. We show that maximum likelihood trajectories are not always well-defined, and describe a polynomial time test for well-definedness. When well-definedness holds, we show that each of the three problems can be solved in polynomial time, and we develop efficient dynamic programming algorithms for doing so.

## 1 Introduction

A continuous-time Markov chain (CTMC) is a model of a dynamical system which, upon entering some state, remains in that state for a random real-valued amount of time (called the dwell time or occupancy time) and then transitions randomly to a new state. CTMCs are used in a wide variety of domains. In stochastic chemical kinetics, states may correspond to the conformation of a molecule such as a protein, peptide or nucleic acid polymer, and transitions correspond to conformational changes (e.g., [1]). Or, the state may correspond to the numbers of different types of molecules in an interacting system, and transitions are the result of chemical reactions between molecules [2]. In phylogenetics, the states may correspond to the genomes of different organisms, and transitions to the evolutionary events (mutations) that separate those organisms [3]. Other application domains include queueing theory, process control and manufacturing, quality control, formal verification, and robot nagivation.

Many computational problems associated with CTMCs have been solved, often by generalizing methods developed for discrete-time Markov chains (DTMCs). For example, stationary distributions for CTMCs can be computed in a manner very similar to that for DTMCs [4]. Estimating the parameters of a CTMC from fully observed data involves estimating state transition probabilities, just as for DTMCs, but adds estimation of the state dwell time distributions. Estimating parameters from partially observed data can be done by a generalization of the well-known Baum-Welch algorithm for parameter estimation for hidden Markov models [5] or by Bayesian methods [6, 7]. When the state of a CTMC is observed periodically through time, but some transitions between observation times may go unseen, the parameter estimation problem can also be solved through embedding

techniques [8]. In scenarios such as manufacturing or robot navigation, one may assume that the state transitions or dwell times are under at least partial control. When control choices are made once for each state entered, dynamic programming and related methods can be used to develop optimal control strategies [9]. When control choices are made continuously in time, methods for hybrid system control are more appropriate [10].

Another fundamental and well-studied problem for CTMCs is to compute, given an initial state and time, the state distribution or most likely state at a later time. These problems are readily solved for DTMCs by dynamic programming [11], but for the CTMCs, solutions have a somewhat different flavor. One approach is based on the forward Chapman-Kolmogorov equations [4], called the Master equation in the stochastic chemical kinetics literature [12]. These specify a system of ordinary differential equations the describe how the probabilities of being in each state change over time. Solving the equations, sometimes analytically but more often numerically, yields the entire state distribution as a function of time. Alternatively, one can uniformize the CTMC, which produces a DTMC along with a probability distribution for a number of transitions to perform. The process obtained by choosing the number of transitions, and then producing a trajectory with that many transitions from the DTMC, has the same state distribution as the original CTMC. This representation allows particularly efficient computation of the state distribution if that distribution is only required at one or a smaller number of different times. Finally, especially in the chemical kinetics community, stochastic simulation algorithms are popular [13]. These approaches act by simply simulating trajectories from the CTMC to produce empirical, numerical estimates of state distributions or other features of the dynamics.

Despite the extensive work on a variety of problems related to to CTMCs, to the best of our knowledge, the problem of finding most likely trajectories has not been addressed. With this paper, we attempt to fill that gap. We propose dynamic programming solutions to three variants of the problem: (i) an initial value problem, where a starting state and final time are given, and we seek the most likely sequence of states and dwell times occurring up until the final time, (ii) a boundary value problem, where initial and final states and times are given, and we seek the most likely intervening trajectory, and (iii) a problem involving partial observability, where we have a sequence of "observations" that may not give full state information, and we want to infer the most likely trajectory that the system followed in producing the observations.

## 2  Definitions

A CTMC is defined by four things: (i) a finite state set $S$, (ii) initial state probabilities, $P_s$ for $s \in S$, (iii) state transition probabilities $P_{ss'}$ for $s, s' \in S$, and (iv) state dwell time parameters $\lambda_s$ for each $s \in S$. Let $S_t \in S$ denote the state of the system at time $t \in [0, +\infty)$. The rules for the evolution of the system are that it starts in state $S_0$, which is chosen according to the distribution $P_s$. At any time $t$, when the system is in state $S_t = s$, the system stays in state $s$ for a random amount of time that is exponentially distributed with parameter $\lambda_s$. When the system finally leaves state $s$, the next state of the system is $s' \neq s$ with probability $P_{ss'}$.

A trajectory of the CTMC is a sequence of states along with the dwell times in all but the last state $\mathbf{U} = (s_0, t_0, s_1, t_1, \ldots, s_{k-1}, t_{k-1}, s_k)$. The meaning of this trajectory is that the system started in state $s_0$, where it stayed for time $t_0$, then transitioned to state $s_1$, where it stayed for time $t_1$, and so on. Eventually, the system reaches state $s_k$, where it remains. Let $\mathbf{U}_t = (s_0, t_0, s_1, t_1, \ldots, s_{k_t-1}, t_{k_t-1}, s_{k_t})$ be a random variable describing the trajectory of the system up until time $t$. In particular, this means that there are $k_t$ state transitions up until time $t$ (where $k_t$ is itself a random variable), the system enters state $s_{k_t}$ sometime at or before time $t$, and remains in state $s_{k_t}$ until sometime after time $t$.

Given the initial state, $S_0$, and a time $t$, the likelihood of a particular trajectory $\mathbf{U}$ is

$$l(\mathbf{U}_t = \mathbf{U}|S_0) = \begin{cases} 0 & \text{if } s_0 \neq S_0 \text{ or } \sum_{i=0}^{k-1} t_i > t \\ \left(\Pi_{i=0}^{k-1} \lambda_{s_i} e^{-\lambda_{s_i} t_i} P_{s_i s_{i+1}}\right) \left(e^{-\lambda_{s_k}\left(t - \sum_i t_i\right)}\right) & \text{otherwise} \end{cases}$$

(1)

When $\sum_i t_i > t$, the likelihood is zero, because it means that the specified transitions have not completed by time $t$. Otherwise, the terms inside the first parentheses account for the likelihood of the dwell times and the state transitions in the sequence, and the term inside the second parentheses

accounts for the probability that the dwell time in the final state does not complete before time $t$. With this notation, the initial value problem we study is easily stated as

$$\arg \max_{\mathbf{U}} l(\mathbf{U}_t = \mathbf{U}|S_0 = s) , \qquad (2)$$

where $s \in S$ and $t > 0$ are both given. The boundary value problem we study is

$$\arg \max_{\mathbf{U}} l(\mathbf{U}_t = \mathbf{U}|S_0 = s, S_t = s'). \qquad (3)$$

Here, the given $s$ and $s'$ are any states in $S$, possibly the same state, and $t > 0$ is also given.

A hidden continuous-time Markov chain (HCTMC) adds an observation model to the CTMC. In particular, we assume a finite set of possible observations $O$. When the system is observed and it is in state $s \in S$, the observer sees observation $o \in O$ with probability $P_{so}$. Let $\mathbf{O} = (o_1, \tau_1, o_2, \tau_2, \ldots, o_m, \tau_m)$ denote a sequence of observations and the times at which they are made. We assume that the observation times are fixed, being chosen ahead of time, and depend in no way on the evolution of the chain itself. Given a trajectory of the system $\mathbf{U} = (s_0, t_0, s_1, t_1, \ldots, t_{k-1}, s_k)$, let $\mathbf{U}(t)$ denote the state of the system at time $t$ implied by that sequence. Then, the probability of an observation sequence $\mathbf{O}$ given the trajectory $\mathbf{U}$ can be written as

$$P(\mathbf{O}|\mathbf{U}_{\tau_m} = \mathbf{U}) = \Pi_{i=1}^m P_{\mathbf{U}(\tau_i)o_i} \qquad (4)$$

The final problem we study in this paper is that of finding the most likely trajectory given an observation sequence:

$$\arg \max_{\mathbf{U}} l(\mathbf{U}_{\tau_m} = \mathbf{U}|\mathbf{O}) \propto \arg \max_{\mathbf{U}} P(\mathbf{O}|\mathbf{U}_{\tau_m} = \mathbf{U})l(\mathbf{U}_{\tau_m} = \mathbf{U}) \qquad (5)$$

## 3  Solving the initial and boundary value problems

In this section we develop solutions to problems (2) and (3). The first step in this development is to show that we can analytically optimize the dwell times if we are given the state sequence. This is covered in the next subsection. Following that, we develop a dynamic program to find optimal state sequences, assuming that the dwell times are set to their optimal values relative to the state sequence.

### 3.1  Maximum likelihood dwell times

Consider a particular trajectory $\mathbf{U} = (s_0, t_0, s_1, t_1, \ldots, s_{k-1}, t_{k-1}, s_k)$. Given $S_0$ and a time $t$, the likelihood of that particular trajectory, $l(\mathbf{U}_t = \mathbf{U}|S_0)$ is given above by Equation (1). Let us assume that $S_0 = s_0$, as we have no need to consider $\mathbf{U}$ starting from the wrong state, and let us maximize $l(\mathbf{U}_t = \mathbf{U}|S_0)$ with respect to the dwell times. To be concise, let $T_{tk} = \{(t_0, t_1, \ldots, t_{k-1}) : t_i \geq 0 \text{ for all } 0 \leq i < k \text{ and } \sum_i t_i \leq t\}$. This is the set of all feasible dwell times for the states up until state $s_k$. Then we can write the desired optimization as

$$\arg \max_{(t_0, \ldots, t_{k-1}) \in T_{tk}} \left( \Pi_{i=0}^{k-1} \lambda_{s_i} e^{-\lambda_{s_i} t_i} P_{s_i s_{i+1}} \right) \left( e^{-\lambda_{s_k}(t - \Sigma_i t_i)} \right) . \qquad (6)$$

It is more convenient to maximize the logarithm, which gives us

$$\arg \max_{(t_0, \ldots, t_{k-1}) \in T_{tk}} \left( \sum_{i=0}^{k-1} \log \lambda_{s_i} - \lambda_{s_i} t_i + \log P_{s_i s_{i+1}} \right) - \lambda_{s_k}(t - \Sigma_j t_j) \qquad (7)$$

Dropping the terms that do not depend on any of the $t_i$ and rearranging, we find the equivalent problem

$$\arg \max_{(t_0, \ldots, t_{k-1}) \in T_{tk}} \sum_{i=0}^{k-1} (\lambda_{s_k} - \lambda_{s_i}) t_i \qquad (8)$$

The solution can be obtained by inspection. If $\lambda_{s_k} \leq \lambda_{s_i}$ for all $0 \leq i < k$, then we must have all $t_i = 0$. That is, the system transitions instantaneously through the states $s_0, s_1, \ldots, s_{k-1}$ and then

dwells in state $s_k$ for (at least) time $t$.[1] Otherwise, let $j$ be such that $\lambda_{s_j}$ is minimal for $0 \leq j < k$. Then an optimal solution has $t_j = t$, and all other $t_i = 0$. Intuitively, this says that if state $s_j$ has the largest expected dwell time (corresponding to the smallest $\lambda$ parameter), then the most likely setting of dwell times is obtained by assuming all of the time $t$ is spent in state $s_j$, and all other transitions happen instantaneously. This is not unintuitive, although it is dissatisfying in the sense that the most likely set of dwell times are not typical in some sense. For example, none are near their expected value. Moreover, the basic character of the solution—that all the time $t$ goes into waiting at the slowest state—is independent of $t$. Nevertheless, being able to solve explicitly for the most likely dwell times for a given state sequence makes it much easier to find the most likely $\mathbf{U}_t$. So, let us press onwards.

## 3.2 Dynamic programming for the most likely state sequence

Substituting back our solution for the $t_i$ into Equation (1), and continuing our assumption that $s_0 = S_0$, we obtain

$$
\max_{(t_0,\ldots,t_{k-1})\in T_{tk}} l(\mathbf{U}_t = \mathbf{U}|S_0) = \begin{cases} \left(\Pi_{i=0}^{k-1} \lambda_{s_i} P_{s_i s_{i+1}}\right) e^{-\lambda_{s_k} t} & \text{if } \lambda_{s_k} \leq \lambda_{s_i} \text{ for} \\ & \text{all } 0 \leq i < k \\ \left(\Pi_{i=0}^{k-1} \lambda_{s_i} P_{s_i s_{i+1}}\right) e^{-(\min_{i=0}^{k-1} \lambda_{s_i})t} & \text{otherwise} \end{cases}
$$
$$
= \left(\Pi_{i=0}^{k-1} \lambda_{s_i} P_{s_i s_{i+1}}\right) e^{-(\min_{i=0}^{k} \lambda_{s_i})t} \tag{9}
$$

This leads to a dynamic program for finding the state sequence that maximizes the likelihood. As is typical, we build maximum likelihood paths of increasing length by finding the best ways of extending shorter paths. The main difference with a more typical scenario is that to score an extension we need to know not just the score and final state of the shorter path, but also the smallest dwell time parameter along that path. Define a $(k, s, \lambda)$-trajectory to be one that includes $k \in \{0, 1, 2, \ldots\}$ state transitions, ends at state $s_k = s$, and for which the smallest dwell time parameter of any state along the trajectory is $\lambda$. Then define $F_k(s, \lambda)$ to be the maximum achievable $l(\mathbf{U}_t = \mathbf{U}|S_0)$, where we restrict attention to $\mathbf{U}$ that are $(k, s, \lambda)$-trajectories. We initialize the dynamic program as:

$$
F_0(S_0, \lambda_{S_0}) = e^{-t\lambda_{S_0}}
$$
$$
F_0(s, \lambda) = 0 \text{ for all } (s, \lambda) \neq (S_0, \lambda_{S_0})
$$

To compute $F_k(s, \lambda)$ for larger $k$, we first observe that $F_k(s, \lambda)$ is undefined if $\lambda > \lambda_s$. This is because there are no $(k, s, \lambda)$-trajectories if $\lambda > \lambda_s$. The fact that a trajectory ends at state $s$ implies that the minimum dwell time parameter along the trajectory can be no greater than $\lambda_s$. So, we only compute $F_k(s, \lambda)$ for $\lambda \leq \lambda_s$.

To determine $F_{k+1}(s, \lambda)$, we must consider two cases. If $\lambda < \lambda_s$, then the best $(k + 1, s, \lambda)$-trajectory must come from some $(k, s', \lambda)$-trajectory. That is, the length $k$ trajectory must already have a dwell time parameter of $\lambda$ along it. The state $s'$ can be any state other than $s$. If $\lambda = \lambda_s$, then the best $(k + 1, s, \lambda)$-trajectory may be an extension of any $(k, s', \lambda')$-trajectory with $\lambda' \geq \lambda$ and $s \neq s'$. To be more concise, define

$$
G(s, \lambda) = \begin{cases} \{\lambda\} & \text{if } \lambda < \lambda_s \\ \{\lambda_{s'} : \lambda_{s'} \geq \lambda\} & \text{if } \lambda = \lambda_s \end{cases} \tag{10}
$$

We then compute $F$ for increasing $k$ as:

$$
F_{k+1}(s, \lambda) = \max_{s' \neq s, \lambda' \in G(s,\lambda)} F_k(s', \lambda') \lambda_{s'} P_{s's} e^{-t(\lambda - \lambda')}
$$

The first term on the right hand side accounts for the likelihood of the best $(k, s', \lambda')$-trajectory. The next two terms account for the dwell in $s'$ and the transition probability to $s$. The final term accounts for any difference between the smallest dwell time parameters along the $k$ and $k + 1$ transition trajectories.

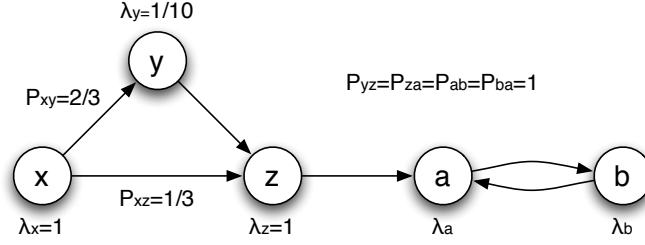

Figure 1: A continuous-time Markov chain used as a demonstration domain. The five circles correspond to states, and the arrows to transitions between states. States are also labeled with their dwell time parameters.

Because the set of possible states, $S$, is finite, so is the set of possible dwell time parameters, $\lambda_s$ for $s \in S$. The size of the table $F_k$ for each $k$ is thus at most $|S|^2$. If we limit $k$ to some maximum value $K$, then the total size of all the tables is at most $K|S|^2$, and the total computational effort $O(K|S|^3)$.

To solve the initial value problem (2), we scan over all values of $k$, $s$ and $\lambda$ to find the maximum value of $F_k(s, \lambda)$. Such a value implies that the most likely state sequence ends at state $s$ after $k$ state transitions. We can use a traceback to reconstitute the full sequence of states, and the result of the previous section to obtain the most likely dwell times. To solve the boundary value problem (3), we do the same, except that we only scan over values of $k$ and $\lambda$, looking for the maximum value of $F_k(S_t, \lambda)$.

## 3.3 Examples

In this section, we use the toy chain depicted in Figure 1 to demonstrate the algorithm of the previous section, and to highlight some properties of maximum likelihood trajectories. First, suppose that we know the system is in state $x$ at time zero and in state $z$ at time $t$. There are two different paths, $(x, z)$ and $(x, y, z)$, that lead from $x$ to $z$. If we ignore the issue of dwell times and consider only the transition probabilities, then the path $(x, y, z)$ seems more probable. Its probability is $P_{xy}P_{yz} = \frac{2}{3} \cdot 1 = \frac{2}{3}$, whereas the direct path $(x, z)$ simply has probability $P_{xz} = \frac{1}{3}$. However, if we consider the dwell times as well, the story can change. For example, suppose that $t = 1$. Note that $\lambda_y = \frac{1}{10}$, so that the expected dwell time in state $y$ is 10. If the chain enters state $y$, the chance of it leaving $y$ before time $t = 1$ is quite small. If we run the dynamic programming algorithm of the previous section to find the most likely trajectory, it finds $(s_0 = x, t_0 = 0, s_1 = z)$ to be most likely, with a score of 0.1226. Along the way, it computes the likelihood of the most likely path going through $y$, which is $(s_0 = x, t_0 = 0, s_1 = y, t_1 = t, s_2 = x)$. It prefers to place all the dwell time $t$ in state $y$, because that state is most likely to have a long dwell time. However, the total score of this trajectory is still only 0.0603, making the direct path the more likely one. On the other hand, if $t = 2$, then the path through $y$ becomes more likely by a score of 0.0546 to 0.0451. If $t = 10$, then the path through $y$ still has a likelihood of 0.0245, whereas the direct path has a likelihood below $2 \times 10^{-5}$, because it is highly unlikely to remain in $x$ and/or $z$ for so long.

Next, suppose that we know $S_0 = a$ and that we are interested in knowing the most likely trajectory out until time $t$, regardless of the final state of that trajectory. For simplicity, suppose also that $\lambda_a = \lambda_b$. There is only one possible state sequence containing $k$ transitions for each $k = 0, 1, 2, \ldots$, and the likelihood of any such sequence turns out to be independent of the dwell times (assuming the dwell times total no more than time $t$):

$$(\Pi_{i=0}^{k-1} \lambda e^{-\lambda t_i}) e^{-\lambda(t - \Sigma_i t_i)} = e^{-\lambda t}\lambda^k \tag{11}$$

If $\lambda < 1$, this implies the optimal trajectory has the system remaining at state $a$. However, if $\lambda = 1$ then all trajectories of all lengths have the same likelihood. If $\lambda > 1$, then there are trajectories of arbitrarily large likelihood, but no maximum likelihood trajectory. Intuitively, because the likelihood of a dwell time can be greater than one, the likelihood of a trajectory can be increased by including short dwells in states with high dwell parameters $\lambda$.

In general, if a continuous-time Markov chain has a cycle of states $(s_0, s_1, \ldots, s_k = s_0)$, such that $\Pi_{i=0}^{k-1} P_{s_i s_{i+1}} \lambda_{s_i} > 1$, then maximum likelihood trajectories do not exist. Rather, a sequence of

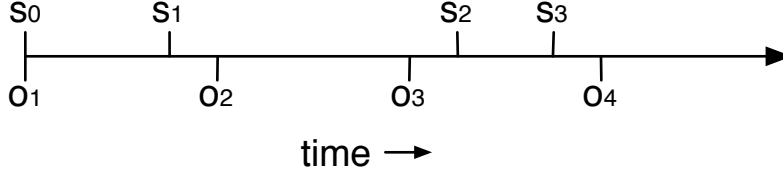

Figure 2: Abstract example of a continuous-time trajectory of a chain, along with observations taken at fixed time intervals.

trajectories with ever-increasing likelihood can be found starting from any state from which the cycle is reachable. One should, thus, always check the chain for this property before seeking maximum likelihood trajectories. This can be easily done in polynomial time. For example, one can label the edges of the transition graph with the weights $\log P_{ss'} \lambda_s$ for the edge from $s$ to $s'$, and then check the graph for the existence of a positive-weight cycle—a well-known polynomial-time computation.

## 4  Solving the partially observable problem

We now turn to problem (12), where we are given an observation sequence $\mathbf{O} = (o_1, \tau_1, o_2, \tau_2, \ldots, o_m, \tau_m)$ and want to find the most likely trajectory $\mathbf{U}$. For simplicity, we assume that $\tau_1 = 0$. The following can be straightforwardly generalized to allow the first observation to take place sometime after the trajectory begins. Similarly, we restrict attention to trajectories $\mathbf{U} = (s_0, t_0, s_1, t_0, \ldots, t_{k-1}, s_k)$ where $\sum_i t_k \leq \tau_m$, so that we do not concern ourselves with extrapolating the trajectory beyond the final observation time. The conditional likelihood of such a trajectory can be written as

$$
\begin{aligned}
l(\mathbf{U}_{\tau_m} = \mathbf{U}|\mathbf{O}) \quad &\propto \quad P(\mathbf{O}|\mathbf{U}_{\tau_m} = \mathbf{U})l(\mathbf{U}_{\tau_m} = \mathbf{U}) && (12) \\
&= \quad \left( \Pi_{i=1}^m P_{\mathbf{U}(\tau_i)} o_i \right) \left( P_{s_0} \left( \Pi_{i=0}^{k-1} \lambda_{s_i} e^{-\lambda_{s_i} t_i} P_{s_i s_{i+1}} \right) \left( e^{-\lambda_{s_k}(t - \Sigma_i t_i)} \right) \right) && (13)
\end{aligned}
$$

The term in the first parentheses is $P(\mathbf{O}|\mathbf{U}_{\tau_m} = \mathbf{U})$, and the term in the second parentheses is $l(\mathbf{U}_{\tau_m} = \mathbf{U})$. The only differences between the second parentheses and Equation (1) is that we now include the probability of starting in state $s_0$, and we have implicitly assumed that $\sum_i t_k \leq \tau_m$, as mentioned above. This form, however, is not convenient for optimizing $\mathbf{U}$. To do this, we need to rewrite $l(\mathbf{U}_{\tau_m} = \mathbf{U})$ in a way that separates the likelihood into events happening in each interval of time between observations.

### 4.1  Decomposing trajectory likelihood by observation intervals

For simplicity, let us further restrict attention to trajectories $\mathbf{U}$ that do not include a transition into a state $s_i$ precisely at any observation time $\tau_j$. We do not have space here to show that this restriction does not affect the value of the optimization problem; this will be addressed in the full paper. The likelihood of the trajectory can be written in terms of the events in each observation interval. For example, consider the trajectory and observations depicted in Figure 2. In the first interval, the system starts in state $s_0$ and transitions to $s_1$, where it stays until time $\tau_2$. The likelihood of this happening is $P_{s_0} \lambda_{s_0} e^{-\lambda_{s_0} t_0} P_{s_0 s_1} e^{-\lambda_{s_1}(\tau_2 - t_0)}$. In the second observation interval, the system never leaves state $s_1$. The probability of this happening is $e^{-\lambda_{s_1}(\tau_3 - \tau_2)}$. Finally, in the third interval, the system continues in state $s_1$ before transitioning to state $s_2$ and then $s_3$, where it remains until the final observation. The likelihood of this happening is $\lambda_{s_1} e^{-\lambda_{s_1}(t_0 + t_1 - \tau_3)} P_{s_1 s_2} \lambda_{s_2} e^{-\lambda_{s_2} t_2} p_{s_2 s_3} e^{-\lambda_{s_3}(\tau_4 - t_0 - t_1 - t_2)}$. If we multiply these together, we obtain the full likelihood of the trajectory, $P_{s_0}(\Pi_{i=0}^2 \lambda_{s_i} e^{-\lambda_{s_i} t_i}) e^{-\lambda_{s_3}(\tau_4 - \Sigma_j t_j)}$.

In general, let $\mathbf{U}_i = (s_{i0}, t_{i0}, s_{i1}, t_{i1}, \ldots, s_{ik_i})$ denote the sequence of states and dwell times of trajectory $\mathbf{U}$ during the time interval $[\tau_i, \tau_{i+1})$. The first dwell time $t_{i0}$, if any, is measured with respect to the start of the time interval. The component of the likelihood of the whole trajectory $\mathbf{U}$ attributable to the $i^{th}$ time interval is nothing other than $l(\mathbf{U}_{\tau_{i+1} - \tau_i} = \mathbf{U}_i | S_0 = s_{i0})$. Thus, the likelihood of the whole trajectory can be written as

$$
l(\mathbf{U}_{\tau_m} = \mathbf{U}) = P_{s_0} \Pi_{i=1}^{m-1} l(\mathbf{U}_{\tau_{i+1} - \tau_i} = \mathbf{U}_i | S_0 = s_{i0}) \tag{14}
$$

## 4.2 Dynamic programming for the optimal trajectory

Combining Equations (12) and (14), we find

$$l(\mathbf{U}_{\tau_m} = \mathbf{U}|\mathbf{O}) \propto P_{\mathbf{U}(0)}P_{\mathbf{U}(0)o_1}\Pi_{i=1}^{m-1}l(\mathbf{U}_{\tau_{i+1}-\tau_i} = \mathbf{U}_i|S_0 = \mathbf{U}(\tau_i))P_{\mathbf{U}(\tau_{i+1})o_{i+1}} \qquad (15)$$

The first two terms account for the probability of the initial state and the probability of the first observation given the initial state. The terms inside the product account for the likelihood of the $i^{th}$ interval of the trajectory, and the probability of the $(i+1)^{st}$ observation, given the state at the end of the $i^{th}$ interval of the trajectory.

One immediate implication of this rewriting of the conditional likelihood is the following. At times $\tau_i$ and $\tau_{i+1}$, the system is in states $\mathbf{U}(\tau_i)$ and $\mathbf{U}(\tau_{i+1})$. If $\mathbf{U}$ is to maximize the conditional likelihood, it had better be that the fragment of the trajectory between those two times, $\mathbf{U}_i$, is a maximum likelihood trajectory from state $\mathbf{U}(\tau_i)$ to state $\mathbf{U}(\tau_{i+1})$ in time $\tau_{i+1} - \tau_i$. If it is not, then an alternative, higher likelihood trajectory fragment could be swapped into $\mathbf{U}$, resulting in a higher conditional likelihood. Let us define

$$H_t(s, s') = \max_{\mathbf{U}'} l(\mathbf{U}_t = \mathbf{U}'|S_0 = s, S_t = s') \qquad (16)$$

to be the maximum achievable likelihood by any trajectory from state $s$ to state $s'$ in time $t$. Then a necessary condition for $\mathbf{U}$ to maximize the conditional likelihood is

$$l(\mathbf{U}_{\tau_{i+1}-\tau_i} = \mathbf{U}_i|S_0 = \mathbf{U}(\tau_i)) = H_{\tau_{i+1}-\tau_i}(\mathbf{U}(\tau_i), \mathbf{U}(\tau_{i+1})) \,. \qquad (17)$$

Moreover, to find an optimal $\mathbf{U}$, we can simply assume that the above condition holds, and concern ourselves only with finding the best endpoints for the each time interval, $\mathbf{U}(\tau_i)$ and $\mathbf{U}(\tau_{i+1})$. (Of course, the endpoint of one interval must be the same as the initial point of the next interval.) Specifically, define $J_i(s)$ to be the likelihood of the most likely trajectory covering the time interval $[\tau_1, \tau_i]$, accounting for the first $i$ observations, and ending at state $s$. The we can compute $J$ as follows. To initialize, we set

$$J_1(s) = P_s P_{so_1} \,. \qquad (18)$$

Then, for $i = 1, 2, \ldots, m-1$,

$$J_{i+1}(s) = \max_{s'} J_i(s')H_{\tau_{i+1}-\tau_i}(s', s)P_{so_{i+1}} \,. \qquad (19)$$

We can then reconstruct the most likely trajectory by finding $s$ that maximizes $J_m(s)$ and tracing back to the beginning. This algorithm is identical to the Viterbi algorithm for finding most likely state sequences for hidden Markov models, with the exception that the state transition probabilities in the Viterbi algorithm are replaced by the $H_{\tau_{i+1}-\tau_i}(s', s)$ terms above—which can, of course, be computed based on the results of the previous section.

## 4.3 Examples

To demonstrate this algorithm, let us return to the CTMC depicted in Figure 1. We assume that $\lambda_a = \lambda_b = 1$, that the system always starts in state $x$, and that when we observe the system, we get a real-valued Gaussian observation with standard deviation 1 and means 0, 10, 3, 100 and 100 for states $x$, $y$, $z$, $a$ and $b$ respectively.[2] The left side of Figure 3 shows three sample sequences of 20 observations. The right side of the figure shows the most likely trajectories inferred under different assumptions. First, if we assume the time interval between observations is $t = 1$, and we consider observations $\mathbf{O}_A$, then the most likely trajectory has the system in state $x$ up through the $10^{th}$ observation, after which it instantly transitions to state $z$ and remains there. This makes sense, as the lower observations at the start of the series are more likely in state $x$. If we consider instead observations $\mathbf{O}_B$, which has a high observation at time $t = 11$, the procedure infers that the system was in state $y$ at that time. Moreover, it predicts that the system switches into $y$ immediately after the $10^{th}$ observation, and says there until just before the $12^{th}$ observation, taking advantage of the fact that longer dwell times are more likely in state $y$ than in the other states. If we consider observations $\mathbf{O}_C$, which have a spike at $t = 5$, the transit to state $y$ is moved earlier, and state $z$ is used to explain observations at $t = 6$ onward, even though the first few are relatively unlikely in that state. If we

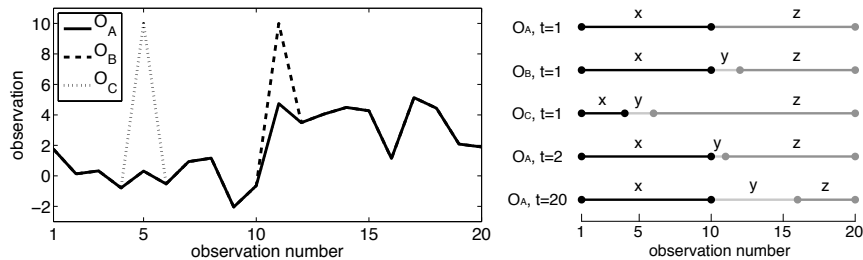

Figure 3: Left: three length-20 observation sequences, $O_A$, $O_B$, and $O_C$. All three are the same at most points, but the $11^{th}$ observation of $O_B$ is 10, and the $5^{th}$ observation of $O_C$ is 10. Right: most likely trajectories inferred by our algorithm, assuming the underlying CTMC is the one given in Figure 1, with parameters given in the text.

return to observations $\mathbf{O}_A$, but we assume that the time interval between observations is $t = 2$, then the most likely trajectory is different than it is for $t = 1$. Although the same states are used to explain the observations, the most likely trajectory has the system transitioning from $x$ to $y$ immediately after the $10^{th}$ observation and dwelling there until just before the $11^{th}$ observation, where the state becomes $z$. This is because, as explained previously, this is the more likely trajectory from $x$ to $z$ given $t = 2$. If we assume the time interval between observations is $t = 20$, then a wider range of observations during the trajectory are attributed to state $y$. Intuitively, this is because, although the observations are somewhat unlikely under state $y$, it is extremely unlikely for the system to dwell for so long in state $z$ as to account for all of the observations from the $11^{th}$ onward.

## 5  Discussion

We have provided correct, efficient algorithms for inferring most likely trajectories of CTMCs given either initial or initial and final states of the chain, or given noisy/partial observations of the chain. Given the enormous practical import of the analogous problems for discrete-time chains, we are hopeful that our methods will prove useful additions to the toolkit of methods available for analyzing continuous-time chains. An alternative, existing approach to the problems we have addressed here is to discretize time, producing a DTMC which is then analyzed by standard methods [14]. A problem with this approach, however, is that if the time step is taken too large, the discretized chain can collapse a whole set of transition sequences of the CTMC into a single "pseudotransition", obscuring the real behavior of the system in continuous time. If the time step is taken to be sufficiently small, then the DTMC should produce substantially the same solutions as our approach. However, the time complexity of the calculations increases as the time step shrinks, which can be a problem if we are interested in long time intervals and/or there are states with very short expected dwell times, necessitating very small time steps.

A related problem on which we are working is to find the most probable state sequence of a continuous-time chain under similar informational assumptions. By this, we mean that the dwell times, rather than being optimized, are marginalized out, so that we are left with only the sequence of states and not the particular times they occurred. In many applications, this state sequence may be of greater interest than the dwell times—especially since, as we have shown, maximum likelihood dwell times are often infinitesimal and hence non-representative of typical system behavior. Moreover, this version of the problem has the advantage of always being well-defined. Because state sequences have probabilities rather than likelihoods, a most probable state sequence will always exist.

## Acknowledgments

Funding for this work was provided in part by the National Sciences and Engineering Research Council of Canada and by the Ottawa Hospital Research Institute.

## Footnotes

[1] If the reader is not comfortable with a dwell time exactly equal to zero, one may instead take $t_i = 0$ as a shorthand for an infinitesimal but positive dwell time. Alternatively, the optimization problem can be modified to explicitly require $t_i > 0$. However, this does nothing to change the fundamental nature of the solution, while resulting in a significantly more laborious exposition.

[2]Although our derivations above assume the observation set $O$ is finite, the same approach goes through if $O$ is continuous and individual observations have likelihoods instead of probabilities.

## References

[1] FG Ball and JA Rice. Stochastic models for ion channels: introduction and bibliography. *Mathematical biosciences*, 112(2):189, 1992.

[2] D.J. Wilkinson. *Stochastic modelling for systems biology*. Chapman & Hall/CRC, 2006.

[3] M. Holder and P.O. Lewis. Phylogeny estimation: traditional and Bayesian approaches. *Nature Reviews Genetics*, 4(4):275–284, 2003.

[4] H.M. Taylor and S. Karlin. *An introduction to stochastic modeling*. Academic Press, 1998.

[5] D.R. Fredkin and J.A. Rice. Maximum likelihood estimation and identification directly from single-channel recordings. *Proceedings: Biological Sciences*, pages 125–132, 1992.

[6] R. Rosales, J.A. Stark, W.J. Fitzgerald, and S.B. Hladky. Bayesian restoration of ion channel records using hidden Markov models. *Biophysical Journal*, 80(3):1088–1103, 2001.

[7] M.A. Suchard, R.E. Weiss, and J.S. Sinsheimer. Bayesian selection of continuous-time Markov chain evolutionary models. *Molecular Biology and Evolution*, 18(6):1001–1013, 2001.

[8] DT Crommelin and E. Vanden-Eijnden. Fitting timeseries by continuous-time Markov chains: A quadratic programming approach. *Journal of Computational Physics*, 217(2):782–805, 2006.

[9] M. L. Puterman. *Markov Decision Processes: Discrete Stochastic Dynamic Programming*. John Wiley and Sons, New York, 1994.

[10] S. Hedlund and A. Rantzer. Optimal control of hybrid systems. In *Decision and Control, 1999. Proceedings of the 38th IEEE Conference on*, volume 4, 1999.

[11] D.P. Bertsekas. *Dynamic programming and optimal control*. Athena Scientific Belmont, Mass, 1995.

[12] NG Van Kampen. *Stochastic processes in physics and chemistry*. North-Holland, 2007.

[13] D. T. Gillespie. Exact stochastic simulation of coupled chemical reactions. *Journal of Physical Chemistry*, 81:2340–2361, 1977.

[14] A. Hordijk, D.L. Iglehart, and R. Schassberger. Discrete time methods for simulating continuous time Markov chains. *Advances in Applied Probability*, pages 772–788, 1976.

